# LEARNING SEQUENTIAL STRUCTURE
# IN SIMPLE RECURRENT NETWORKS

David Servan-Schreiber, Axel Cleeremans, and James L. McClelland
Departments of Computer Science and Psycholgy
Carnegie Mellon University
Pittsburgh, PA 15213

## ABSTRACT

We explore a network architecture introduced by Elman (1988) for predicting successive elements of a sequence. The network uses the pattern of activation over a set of hidden units from time-step t-1, together with element t, to predict element t+1. When the network is trained with strings from a particular finite-state grammar, it can learn to be a perfect finite-state recognizer for the grammar. Cluster analyses of the hidden-layer patterns of activation showed that they encode prediction-relevant information about the entire path traversed through the network. We illustrate the phases of learning with cluster analyses performed at different points during training.

Several connectionist architectures that are explicitly constrained to capture sequential information have been developed. Examples are Time Delay Networks (e.g, Sejnowski & Rosenberg, 1986) -- also called 'moving window' paradigms -- or algorithms such as back-propagation in time (Rumelhart, Hinton & Williams, 1986). Such architectures use explicit representations of several consecutive events, if not of the entire history of past inputs. Recently, Elman (1988) has introduced a simple recurrent network (SRN) that has the potential to master an infinite corpus of sequences with the limited means of a learning procedure that is *completely local in time* (see Figure 1.).

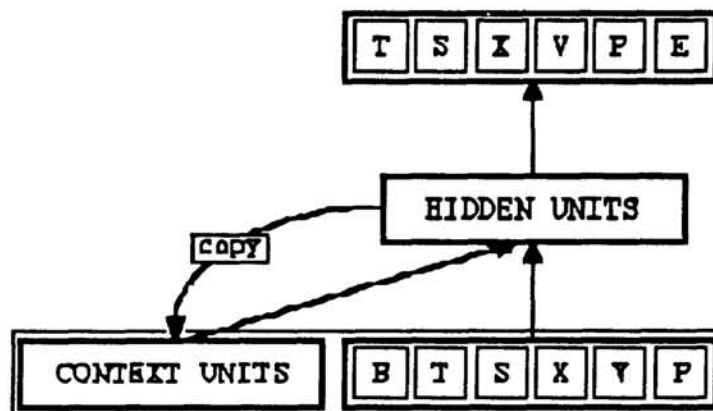

**Figure 1.** The simple recurrent network (Elman, 1988)

In the SRN, the pattern of activation on the hidden units at time **t-1**, together with the new input pattern, is allowed to influence the pattern of activation at time **t** . This is achieved by *copying* the pattern of activation on the hidden layer at time **t-1** to a set of input units -- called the 'context units' -- at time **t**. The forward connections in the network are subject to training via back-propagation. but there is no backpropagation through time.

In this paper, we show that the SRN can learn to mimic closely a finite state automaton, both in its behavior and in its state representations. In particular, we show that it can learn to process an *infinite* corpus of strings based on experience with a *finite* set of training exemplars. We then describe the phases through which the appropriate internal representations are discovered during training.

## MASTERING A FINITE STATE GRAMMAR

In our first experiment, we asked whether the network could learn the contingencies implied by a small finite state grammar (see Figure 2). The network was presented with strings derived from this grammar, and was required to try to predict the next letter at every step. These predictions are context dependent since each letter appears twice in the grammar and is followed in each case by different successors.

A single unit on the input layer represented a given letter (six input units in total; five for the letters and one for a begin symbol 'B'). Similar local representations were used on the output layer (with the 'begin' symbol being replaced by an end symbol 'E'). There were three hidden units.

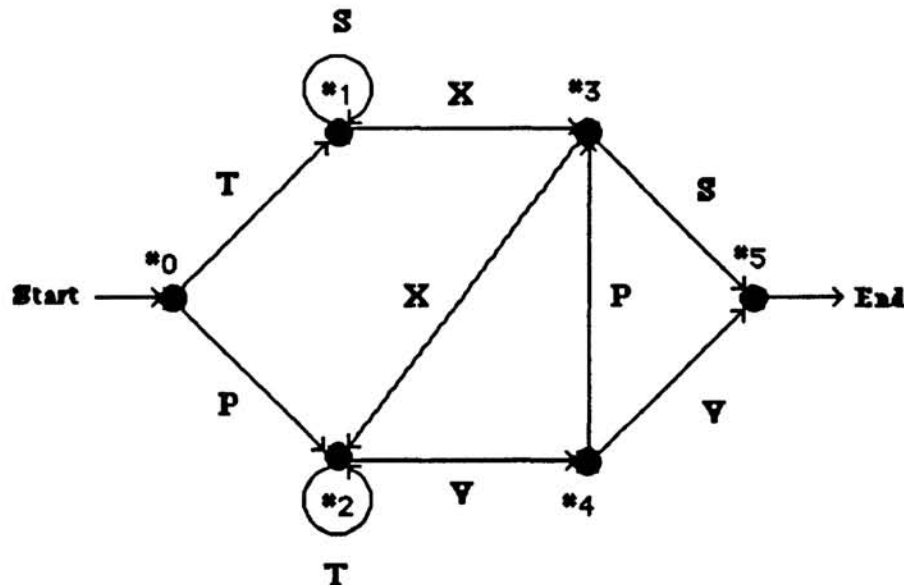

**Figure 2.** The small finite-state grammar (Reber, 1967)

*Training.* On each of 60,000 training trials, a string was generated from the grammar, starting with 'B'. Successive arcs were selected randomly from the 2 possible continuations with a probability of 0.5. Each letter was

then presented sequentially to the network. The activations of the context units were reset to 0.5 at the beginning of each string. After each letter, the error between the network's prediction and the *actual successor* specified by the string was computed and back-propagated. The 60,000 randomly generated strings ranged from 3 to 30 letters (mean: 7; sd: 3.3).

*Performance.* Three tests were conducted. First, we examined the network's predictions on a set of 70,000 random strings. During this test, the network is first presented with the start signal, and one of the five letters or E is then selected at random as a successor. If that letter is predicted by the network as a legal successor (i.e, activation is above 0.3 for the corresponding unit), it is then presented to the input layer on the next time step, and another letter is drawn at random as its successor. This procedure is repeated as long as each letter is predicted as a legal successor  until the end signal is selected as the next letter. The procedure is interrupted as soon as the actual successor generated by the random procedure is not predicted by the network, and the string of letters is then considered 'rejected'. A string is considered 'accepted' if all its letters have been predicted as possible continuations up to, and including, the end signal. Of the 70,000 random strings, 0.3 % were grammatical, and 99.7 % were ungrammatical. The network performed flawlessly, accepting all the grammatical strings and rejecting all the others. In a second test, we presented the network with 20,000 generated at random *from the grammar*, i.e, all these strings were grammatical. Using the same criterion as above, all of these strings were correctly 'accepted'. Finally, we constructed a set of very long grammatical strings -- more than 100 letters long -- and verified that at each step the network correctly predicted *all* the possible successors (activations above 0.3) and *none* of the other letters in the grammar.

*Analysis of internal representations.*   What kind of internal representations have developed over the set of hidden units that allow the network to associate the proper predictions to intrinsically ambiguous letters? One way to answer this question is to record the hidden units' activation patterns generated in response to the presentation of individual letters in different contexts. These activation vectors can then be used as input to a cluster analysis program. Figure 3.A. shows the results of such an analysis conducted on a small random set of grammatical strings. The patterns of activation are grouped according to the nodes of the grammar: all the patterns that are used to predict the successors of a given node are grouped together independently of the current letter. This observation sheds some light on the behavior of the network: at each point in a sequence, the pattern of activation stored over the context units provides information about the current node in the grammar. Together with information about the current letter (represented on the input layer), this contextual information is used to produce a new pattern of activation over the hidden layer, that uniquely specifies the *next* node. In that sense, the network closely approximates the finite-state automaton that would encode the grammar from which the training exemplars were derived. However, a closer look at the cluster analysis reveals that within a cluster corresponding to a particular node, patterns are further divided according to the path traversed before the node is reached. For example, looking at the bottom cluster -- node #5 -- patterns produced by a 'VV', 'PS', 'XS' or 'SXS' ending are grouped separately:

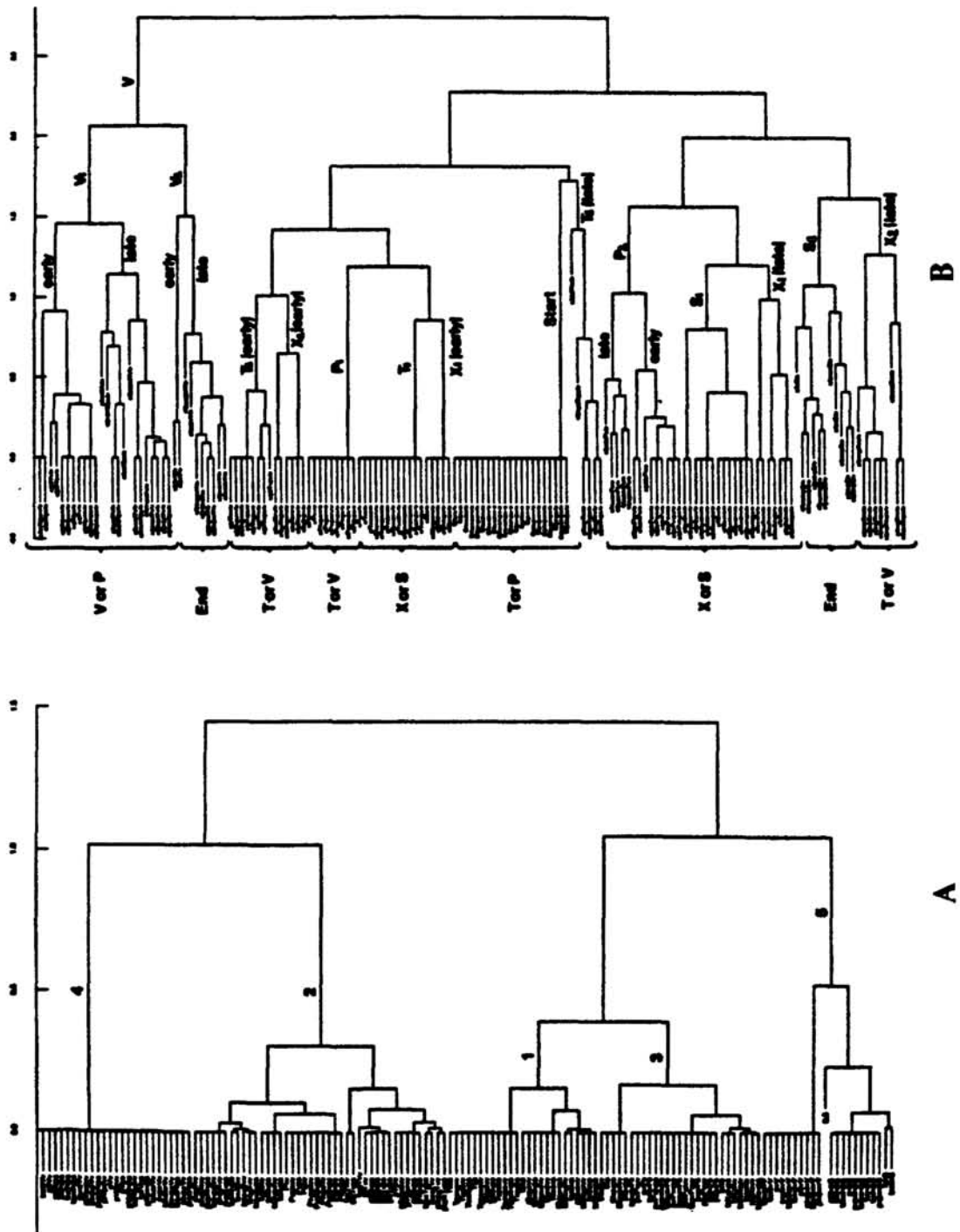

**Figure 3. A.** Hierarchical cluster analysis of the hidden unit activation patterns after 60,000 presentations of strings generated at random from the finite-state grammar. **B.** Cluster analysis of the H.U. activation patterns following 2000 epochs of training on a set of 22 strings with a maximum length of eight letters.

they are more similar to each other than to the abstract prototype of node #5. This tendency to preserve information about the path is not a characteristic of traditional finite-state automata.

## ENCODING PATH INFORMATION

In a different set of experiments, we asked whether the SRN could learn to use the information about the path that is encoded in the hidden units' patterns of activation. In one of these experiments, we tested whether the network could master length constraints. When strings generated from the small finite-state grammar may only have a maximum of 8 letters, the prediction following the presentation of the same letter in position number six or seven may be different. For example, following the sequence 'TSSSXXV', 'V' is the seventh letter and only another 'V' would be a legal successor. In contrast, following the sequence 'TSSXXV', both 'V' and 'P' are legal successors. A network with 15 hidden units was trained on a small set of length-limited (max. 8 letters) grammatical strings. It was able to use the small activation differences present over the context units - and due to the slightly different sequences presented - to master contingencies such as those illustrated above (see table 1).

**Table 1.** Activation of each output unit following the presentation of 'V' as the 6th or 7th letter in the string

|        | T   | S   | P    | X   | V    | E   |
|--------|-----|-----|------|-----|------|-----|
| tssxxV | 0.0 | 0.0 | 0.54 | 0.0 | 0.48 | 0.0 |
| tsssxxV| 0.0 | 0.0 | 0.02 | 0.0 | 0.97 | 0.0 |

A cluster analysis of all the patterns of activation on the hidden layer generated by each letter in each sequence demonstrates how the influence of the path is reflected in these patterns (see figure 3.B.)[*]. We labeled the arcs according to the letter being presented (the 'current letter') and its position in the grammar defined by Reber. Thus 'V$_1$' refers to the first 'V' in the grammar and 'V$_2$' to the second 'V' which immediately precedes the end of the string. 'Early' and 'Late' refer to whether the letter occurred early or late in the sequence (for example in 'PT..' 'T$_2$' occurs early; in 'PVPXT..' it occurs late). Finally, in the left margin we indicated what predictions the corresponding patterns yield on the output layer (e.g, the hidden unit pattern generated by 'BEGIN' predicts 'T' or 'P').

From the figure, it can be seen that the patterns are grouped according to three distinct principles: (1) according to similar predictions, (2) according to similar letters presented on the input units, and (3) according to similar paths. These factors do not necessarily overlap since several occurrences of the same letter in a sequence usually implies different predictions and since similar paths also lead to different predictions depending on the current letter. For example, the top cluster in the figure corresponds to all occurrences of the letter 'V' and is further subdivided among 'V$_1$' and 'V$_2$'.

---

[*] Information about the leaves of the cluster analyses in this and the remaining figures is available in Servan-Schreiber, Cleeremans and McClelland (1988).

The 'V$_1$' cluster is itself further divided between groups where 'V$_1$' occurs early in the sequence (e.g, 'pV...') and groups where it occurs later (e.g, 'tssxxV...' and 'pvpxV...'). Note that the division according to the path does not necessarily correspond to different predictions. For example, 'V$_2$' always predicts 'END' and always with maximum certainty. Nevertheless, sequences up to 'V$_2$' are divided according to the path traversed.

## PHASES OF LEARNING

How can information about the path be progressively encoded in the hidden layer patterns of activation? To clarify how the network learns to use the context of preceding letters in a sequence, we will illustrate the different phases of learning with cluster analyses of the hidden layer patterns generated at each phase. To make the analyses simpler, we used a smaller training set than the training set mentioned previously. The corresponding finite-state grammar is shown in Figure 4. In this simpler grammar, the main difference -- besides the reduced number of patterns -- is that the letters 'P' and 'T' appear only once.

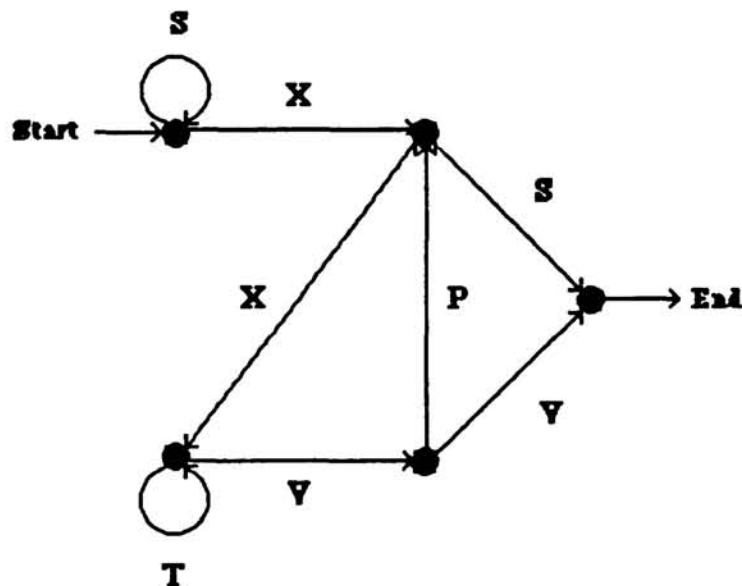

**Figure 4.** The reduced finite-state grammar from which 12 strings were generated for training

*Discovering letters.* At epoch 0, before the network has received any training, the hidden unit patterns clearly show an organization by letter: to each letter corresponds an individual cluster. These clusters are already subdivided according to preceding sequences -- the 'path'. This fact illustrates how a pattern of activation on the context units naturally tends to encode the path traversed so far independently of any error correcting procedure. The average distance between the different patterns -- the 'contrast' as it were -- is nonetheless rather small; the scale only goes up to 0.6 (see Figure 5.A.)**. But this is due to the very small initial random

** In all the following figures, the scale was automatically determined by the cluster analysis program. It is important to keep this in mind when comparing the figures to

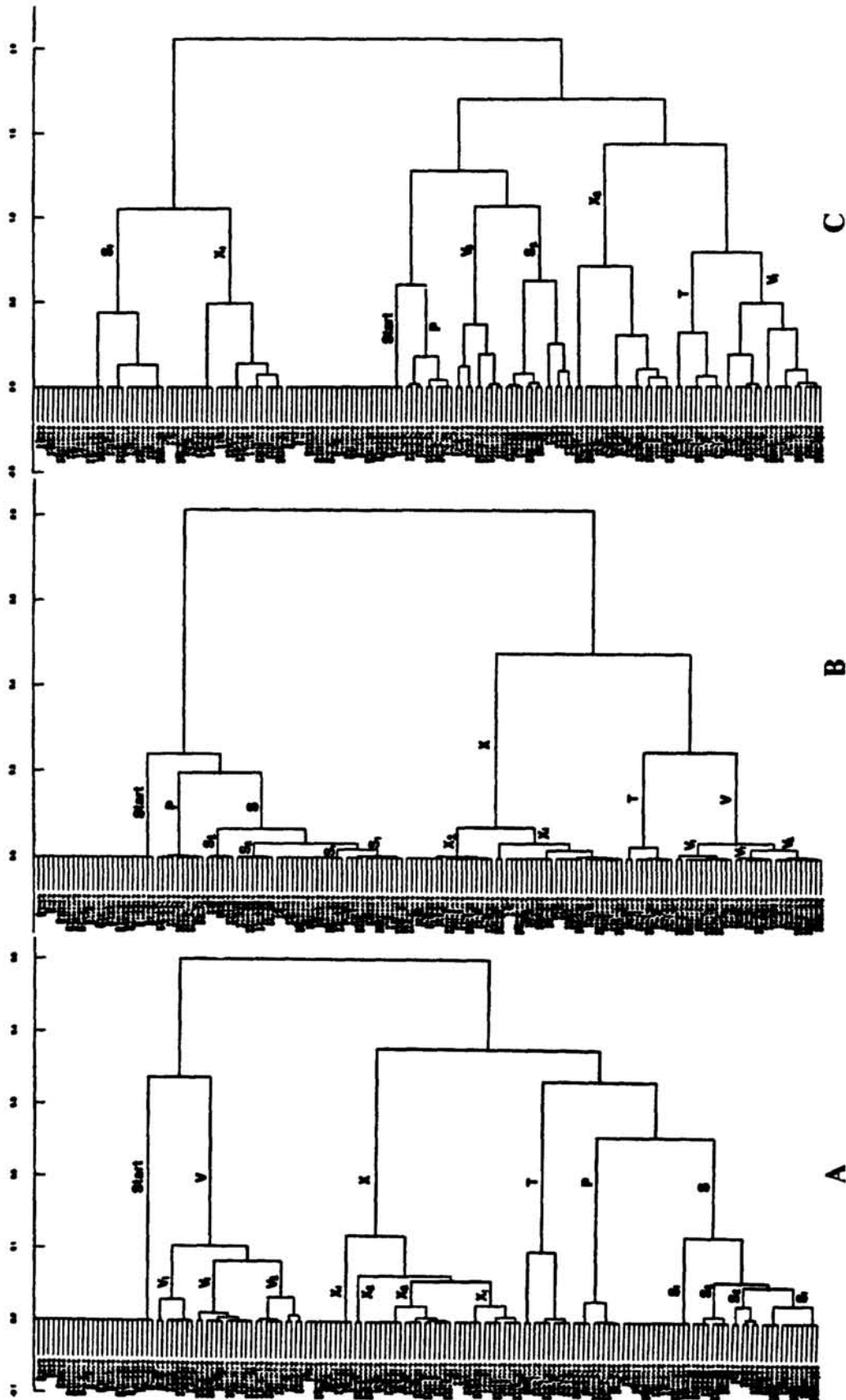

**Figure 5.** Cluster Analyses of the H.U. activation patterns obtained with the reduced set of strings: **A.** before training. **B.** After 100 epochs of training. **C.** After 700 epochs of training.

values of the weights from the input and context layers to the hidden layer. Larger initial values would enhance the network's tendency to capture path information in the hidden unit patterns before training is even started.

After 100 epochs of training, an organization by letters is still prevalent, however letters have been regrouped according to similar predictions. 'START', 'P' and 'S' all make the common prediction of 'X or S' (although 'S' also predicts 'END'); 'T' and 'V' make the common prediction of 'V' (although 'V' also predicts 'END' and 'P'). The path information has been almost eliminated: there is very little difference between the patterns generated by two different occurrences of the same letter (see Figure 5.B.). For example, the hidden layer pattern generated by 'S$_1$' and the corresponding output pattern are almost identical to the patterns generated by 'S$_2$' (see table 2).

**Table 2.** Activation of each output unit following the presentation of the first S in the grammar (S$_1$) or the second S (S$_2$) after 100 epochs of training

|    | T | S | P | X | V | E |
|----|---|---|---|---|---|---|
| S1 | 0.0 | 0.36 | 0.0 | 0.33 | 0.16 | 0.17 |
| S2 | 0.0 | 0.37 | 0.0 | 0.33 | 0.16 | 0.17 |

In this phase, the network is learning to ignore the pattern of activation on the context units and to produce an output pattern appropriate to the letter 'S' in any context. This is a direct consequence of the fact that the patterns of activation on the hidden layer -- and hence the context layer -- are continuously changing from one epoch to the next as the weights from the input units (the letters) to the hidden layer are modified. Consequently, adjustments made to the weights from the context layer to the hidden layer are inconsistent from epoch to epoch and cancel each other. In contrast, the network is able to pick up the stable association between each letter and all of its possible successors.

*Discovering arcs.* At the end of this phase, individual letters consistently generate a unique pattern of activation on the hidden layer. This is a crucial step in developing a sensitivity to context: patterns copied onto the context layer have become a unique code designating which letter immediately preceded the current letter. The learning procedure can now exploit the regular association between the pattern on the context layer and the desired output. Around epoch 700, the cluster analysis shows that the network has used this information to differentiate clearly between the first and second occurrence of the same letter (Figure 5.C.). The pattern generated by 'S$_2$' -- which predicts 'END' -- clusters with the pattern generated by 'V$_2$', which also predicts 'END'. The overall difference between all the hidden layer patterns has also more than roughly doubled, as indicated by the change in scale.

*Encoding the path.* During the last phase of learning, the network learns to make different predictions to the same occurrence of a letter (e.g, 'V$_1$')

each other.

on the basis of the previous sequence. For example, it learns to differentiate between 'ssxxV' which predicts either 'P' or 'V', and 'sssxxV' which predicts only 'V' by exploiting the small difference between the activation patterns generated by $X_2$ in the two different contexts.

The process through which path information is encoded can be conceptualized in the following way: As the initial papers about back-propagation pointed out, the hidden unit patterns of activation represent an 'encoding' of the features of the input patterns that are relevant to the task. In the recurrent network, the hidden layer is presented with information about the current letter, but also -- on the context layer -- with an encoding of the relevant features of the previous letter. Thus, a given hidden layer pattern can come to encode information about the relevant features of two consecutive letters. When this pattern is fed back on the context layer, the new pattern of activation over the hidden units can come to encode information about three consecutive letters, and so on. In this manner, the context layer patterns can allow the network to maintain prediction-relevant features of an entire sequence. However, it is important to note that information about the path that is not *relevant locally* (i.e, that does not contribute to predicting successors of the *current* letter) tends not to be encoded in the next hidden layer pattern. It may then be lost for subsequent processing. This tendency is lessened when the network has extra degrees of freedom -- i.e, extra hidden units -- so as to allow small and locally useless differences to survive for several processing steps.

## CONCLUSION

We have shown that the network architecture first proposed by Elman (1988) is capable of mastering an infinite corpus of strings generated from a finite-state grammar after training on a finite set of exemplars with a learning algorithm that is local in time. The network develops internal representations that correspond to the nodes of the grammar and closely approximates the corresponding minimal finite-state recognizer. We have also shown that the simple recurrent network is able to encode information about contingencies that are not local to a given letter and its immediate predecessor, such as those implied by a length constraint on the strings. Encoding of sequential structure in the patterns of activation over the hidden layers proceeds in stages. The network first develops stable hidden-layer representations for individual letters, and then for individual arcs in the grammar. Finally, the network is able to exploit slight differences in the patterns of activation which denote a specific path through the grammar. Our current work is exploring the relevance of this architecture to the processing of embedded sequences typical of natural language. The results of some preliminary experiments are available in Servan-Schreiber, Cleeremans and McClelland (1988).

## References

Elman, J.L. (1988). Finding structure in time. CRL Technical report 9901. Center for Research in Language, University of California, San Diego.

Reber, A.S. (1967). Implicit learning of artificial grammars. *Journal of Verbal Learning and Verbal Behavior*, 5, 855-863.

Rumelhart, D.E., Hinton, G.E., and Williams, R.J. (1986). Learning internal representations by backpropagating errors. *Nature* 323:533-536.

Sejnowski, T.J. and Rosenberg C. (1986). NETtalk: A parallel network that learns to read aloud. Technical Report, Johns Hopkins University JHU-EECS-86-01.

Servan-Schreiber D, Cleeremans A, and McClelland JL (1988) Encoding sequential structure in simple recurrent networks. Technical Report CMU-CS-88-183, Computer Science Department, Carnegie Mellon University, Pittsburgh, PA 15213.

Williams, R.J. and Zipser, D. (1988). A learning algorithm for continually running fully recurrent neural networks. ICS Technical report 8805. Institute for Cognitive Science, UCSD, La Jolla, CA 92093.
